# Online Learning: Random Averages, Combinatorial Parameters, and Learnability

**Alexander Rakhlin**
Department of Statistics
University of Pennsylvania

**Karthik Sridharan**
Toyota Technological Institute
at Chicago

**Ambuj Tewari**
Computer Science Department
University of Texas at Austin

## Abstract

We develop a theory of online learning by defining several complexity measures. Among them are analogues of Rademacher complexity, covering numbers and fat-shattering dimension from statistical learning theory. Relationship among these complexity measures, their connection to online learning, and tools for bounding them are provided. We apply these results to various learning problems. We provide a complete characterization of online learnability in the supervised setting.

## 1  Introduction

In the online learning framework, the learner is faced with a sequence of data appearing at discrete time intervals. In contrast to the classical "batch" learning scenario where the learner is being evaluated after the sequence is completely revealed, in the online framework the learner is evaluated at every round. Furthermore, in the batch scenario the data source is typically assumed to be $i.i.d.$ with an unknown distribution, while in the online framework we relax or eliminate any stochastic assumptions on the data source. As such, the online learning problem can be phrased as a repeated two-player game between the learner (player) and the adversary (Nature).

Let $\mathcal{F}$ be a class of functions and $\mathcal{X}$ some set. The `Online Learning Model` is defined as the following $T$-round interaction between the learner and the adversary: On round $t = 1, \ldots, T$, the Learner chooses $f_t \in \mathcal{F}$, the Adversary picks $x_t \in \mathcal{X}$, and the Learner suffers loss $f_t(x_t)$. At the end of $T$ rounds we define *regret* as the difference between the cumulative loss of the player as compared to the cumulative loss of the best fixed comparator. For the given pair $(\mathcal{F}, \mathcal{X})$, the problem is said to be *online learnable* if there exists an algorithm for the learner such that regret grows sublinearly. Learnability is closely related to *Hannan consistency* [13, 9].

There has been a lot of interest in a particular setting of the online learning model, called *online convex optimization*. In this setting, we write $x_t(f_t)$ as the loss incurred by the learner, and the assumption is made that the function $x_t$ is convex in its argument. The particular convexity structure enables the development of optimization-based algorithms for learner's choices. Learnability and precise rates of growth of regret have been shown in a number of recent papers (e.g. [33, 25, 1]). The online learning model also subsumes the *prediction* setting. In the latter, the learner's choice of a $\mathcal{Y}$-valued function $g_t$ leads to the loss of $\ell(g_t(z_t), y_t)$ according to a fixed loss function $\ell : \mathcal{Y} \times \mathcal{Y} \mapsto \mathbb{R}$. The choice of the learner is equivalently written as $f_t(x) = \ell(g_t(z), y)$, and $x_t = (z_t, y_t)$ is the choice of the adversary. In Section 6 we discuss the prediction setting in more detail.

In the "batch" learning scenario, data $\{(x_i, y_i)\}_{i=1}^{T}$ is presented as an i.i.d. draw from a fixed distribution over some product $\mathcal{X} \times \mathcal{Y}$. Learnability results have been extensively studied in the PAC framework [29] and its agnostic extensions [14, 17]. It is well-known that learnability in the binary case (that is, $\mathcal{Y} = \{-1, +1\}$) is completely characterized by finiteness of the Vapnik-Chervonenkis combinatorial dimension of the function class [32, 31]. In the real-valued case, a number of combinatorial quantities have been proposed: $P$-dimension [23], $V$-dimension, as well as the *scale-sensitive* versions $P_\gamma$-dimension [17, 5] and $V_\gamma$-dimension [3]. The last two dimensions

were shown to be characterizing learnability [3] and uniform convergence of means to expectations for function classes.

In contrast to the classical learning setting, there has been surprisingly little work on characterizing learnability for the online learning framework. Littlestone [19] has shown that, in the setting of prediction of binary outcomes, a certain combinatorial property of the binary-valued function class characterizes learnability in the realizable case. The result has been extended to the non-realizable case by Shai Ben-David, Dávid Pál and Shai Shalev-Shwartz [7] who named this combinatorial quantity the *Littlestone's dimension*. In parallel to [7], minimax analysis of online convex optimization yielded new insights into the value of the game, its minimax dual representation, as well as algorithm-independent upper and lower bounds [1, 27]. In this paper, we build upon these results and the findings of [7] to develop a theory of online learning.

We show that in the online learning model, a notion which we call *Sequential Rademacher complexity* allows us to easily prove learnability for a vast array of problems. The role of this complexity is similar to the role of the Rademacher complexity in statistical learning theory. Next, we extend Littlestone's dimension to the real-valued case. We show that finiteness of this scale-sensitive version, which we call the *fat-shattering dimension*, is necessary and sufficient for learnability in the prediction setting. Extending the binary-valued result of [7], we introduce a generic algorithm which plays the role similar to that of empirical risk minimization for i.i.d. data: if the problem is learnable in the supervised setting, then it is learnable by this algorithm. Along the way we develop analogues of Massart's finite class lemma, the Dudley integral upper bound on the Sequential Rademacher complexity, appropriately defined packing and covering numbers, and even an analogue of the Sauer-Shelah combinatorial lemma. In the full version of this paper, we introduce a generalization of the uniform law of large numbers for non-i.i.d. distributions and show that finiteness of the fat-shattering dimension implies this convergence.

Many of the results come with more work than their counterparts in statistical learning theory. In particular, instead of training sets we have to work with trees, making the results somewhat involved. For this reason, we state our results without proofs, deferring the details to the full version of this paper. While the spirit of the online theory is that it provides a "temporal" generalization of the "batch" learning problem, not all the results from statistical learning theory transfer to our setting. For instance, two distinct notions of a packing set exist for trees, and these notions can be seen to coincide in "batch" learning. The fact that many notions of statistical learning theory can be extended to the online learning model is indeed remarkable.

## 2 Preliminaries

By phrasing the online learning model as a repeated game and considering its minimax value, we naturally arrive at an important object in combinatorial game theory: trees. Unless specified, all trees considered in this paper are *rooted binary* trees with equal-depth paths from the root to the leaves. While it is useful to have the tree picture in mind when reading the paper, it is also necessary to precisely define trees as mathematical objects. We opt for the following definition. Given some set $\mathcal{Z}$, a $\mathcal{Z}$-*valued tree of depth* $T$ is a sequence $(\mathbf{z}_1, \ldots, \mathbf{z}_T)$ of $T$ mappings $\mathbf{z}_i : \{\pm 1\}^{i-1} \mapsto \mathcal{Z}$. The *root* of the tree $\mathbf{z}$ is the constant function $\mathbf{z}_1 \in \mathcal{Z}$. Armed with this definition, we can talk about various operations on trees. For a function $f : \mathcal{Z} \mapsto \mathcal{U}$, $f(\mathbf{x})$ denotes the $\mathcal{U}$-valued tree defined by the mappings $(f \circ \mathbf{x}_1, \ldots, f \circ \mathbf{x}_T)$. A *path* of length $T$ is a sequence $\epsilon = (\epsilon_1, \ldots, \epsilon_{T-1}) \in \{\pm 1\}^{T-1}$. We shall abuse notation by referring to $\mathbf{x}_i(\epsilon_1, \ldots, \epsilon_{i-1})$ by $\mathbf{x}_i(\epsilon)$. Clearly $\mathbf{x}_i$ only depends on the first $i - 1$ elements of $\epsilon$.

We denote $(y_a, \ldots, y_b)$ by $y_{a:b}$. The set of all functions from $\mathcal{X}$ to $\mathcal{Y}$ is denoted by $\mathcal{Y}^{\mathcal{X}}$, and the $t$-fold product $\mathcal{X} \times \ldots \times \mathcal{X}$ is denoted by $\mathcal{X}^t$. For any $T \in \mathbb{N}$, $[T]$ denotes the set $\{1, \ldots, T\}$. Whenever the variable in $\sup$ ($\inf$) is not quantified, it ranges over the set of all possible values.

## 3 Value of the Game

Fix the sets $\mathcal{F}$ and $\mathcal{X}$ and consider the online learning model stated in the introduction. We assume that $\mathcal{F}$ is a separable metric space. Let $\mathcal{Q}$ be the set of Borel probability measures on $\mathcal{F}$. Assume that $\mathcal{Q}$ is weakly compact. We consider randomized learners who predict a distribution $q_t \in \mathcal{Q}$ on every

round. Formally, define a learner's strategy $\pi$ as a sequence of mappings $\pi_t : \mathcal{X}^{t-1} \times \mathcal{F}^{t-1} \mapsto \mathcal{Q}$ for each $t \in [T]$. We define the value of the game as

$$\mathcal{V}_T(\mathcal{F}, \mathcal{X}) = \inf_{q_1 \in \mathcal{Q}} \sup_{x_1 \in \mathcal{X}} \mathbb{E}_{f_1 \sim q_1} \cdots \inf_{q_T \in \mathcal{Q}} \sup_{x_T \in \mathcal{X}} \mathbb{E}_{f_T \sim q_T} \left[ \sum_{t=1}^{T} f_t(x_t) - \inf_{f \in \mathcal{F}} \sum_{t=1}^{T} f(x_t) \right] \quad (1)$$

where $f_t$ has distribution $q_t$. We consider here the *adaptive* adversary who gets to choose each $x_t$ based on the history of moves $f_{1:t-1}$ and $x_{1:t-1}$.

Note that our assumption that $\mathcal{F}$ is a separable metric space implies that $\mathcal{Q}$ is tight [28] and Prokhorov's theorem states that compactness of $\mathcal{Q}$ under weak topology is equivalent to tightness [28]. Hence we have that $\mathcal{Q}$ is compact under weak topology and this is essentially what we need to apply a modification of Theorem 1 of [1]. Specifically we show the following:

**Theorem 1.** *Let $\mathcal{F}$ and $\mathcal{X}$ be the sets of moves for the two players, satisfying the necessary conditions for the minimax theorem to hold. Denote by $\mathcal{Q}$ and $\mathcal{P}$ the sets of probability distributions (mixed strategies) on $\mathcal{F}$ and $\mathcal{X}$, respectively. Then*

$$\mathcal{V}_T(\mathcal{F}, \mathcal{X}) = \sup_{p_1} \mathbb{E}_{x_1 \sim p_1} \ldots \sup_{p_T} \mathbb{E}_{x_T \sim p_T} \left[ \sum_{t=1}^{T} \inf_{f_t \in \mathcal{F}} \mathbb{E}_{x_t \sim p_t} \left[ f_t(x_t) \right] - \inf_{f \in \mathcal{F}} \sum_{t=1}^{T} f(x_t) \right]. \quad (2)$$

The question of learnability in the online learning model is now reduced to the study of $\mathcal{V}_T(\mathcal{F}, \mathcal{X})$, taking Eq. (2) as the starting point. In particular, under our definition, showing that the value grows sublinearly with $T$ is equivalent to showing learnability.

**Definition 1.** A class $\mathcal{F}$ is said to be *online learnable* with respect to the given $\mathcal{X}$ if

$$\limsup_{T \to \infty} \frac{\mathcal{V}_T(\mathcal{F}, \mathcal{X})}{T} = 0 .$$

The rest of the paper is aimed at understanding the value of the game $\mathcal{V}_T(\mathcal{F}, \mathcal{X})$ for various function classes $\mathcal{F}$. Since complexity of $\mathcal{F}$ is the focus of the paper, we shall often write $\mathcal{V}_T(\mathcal{F})$, and the dependence on $\mathcal{X}$ will be implicit. One of the key notions introduced in this paper is the complexity which we term *Sequential Rademacher complexity*. A natural generalization of Rademacher complexity [18, 6, 21], the sequential analogue possesses many of the nice properties of its classical cousin. The properties are proved in Section 7 and then used to show learnability for many of the examples in Section 8. The first step, however, is to show that Sequential Rademacher complexity upper bounds the value of the game. This is the subject of the next section.

## 4  Random Averages

**Definition 2.** The *Sequential Rademacher Complexity* of a function class $\mathcal{F} \subseteq \mathbb{R}^{\mathcal{X}}$ is defined as

$$\mathfrak{R}_T(\mathcal{F}) = \sup_{\mathbf{x}} \mathbb{E}_{\epsilon} \left[ \sup_{f \in \mathcal{F}} \sum_{t=1}^{T} \epsilon_t f(\mathbf{x}_t(\epsilon)) \right]$$

where the outer supremum is taken over all $\mathcal{X}$-valued trees of depth $T$ and $\epsilon = (\epsilon_1, \ldots, \epsilon_T)$ is a sequence of i.i.d. Rademacher random variables.

**Theorem 2.** *The minimax value of a randomized game is bounded as $\mathcal{V}_T(\mathcal{F}) \leq 2\mathfrak{R}_T(\mathcal{F})$ .*

Theorem 2 relies on a technical lemma, whose proof requires considerably more work than the classical symmetrization proof [11, 21] due to the non-i.i.d. nature of the sequences. We mention that under strong assumptions on the space of functions, the Sequential Rademacher and the classical Rademacher complexities coincide (see [1]). In general, however, the two complexities are very different. For example, the discrepancy is exhibited by a class of linear threshold functions.

## 5  Covering Numbers and Combinatorial Parameters

In online learning, the notion characterizing learnability for binary prediction in the realizable case has been introduced by Littlestone [19] and extended to the non-realizable case of binary prediction by Shai Ben-David, Dávid Pál and Shai Shalev-Shwartz [7]. Next, we define the Littlestone's

dimension [19, 7] and propose its scale-sensitive versions for real-valued function classes. In the sequel, these combinatorial parameters are shown to control the growth of covering numbers on trees. In the setting of prediction, the combinatorial parameters are shown to exactly characterize learnability (see Section 6).

**Definition 3** ([19, 7]). *An $\mathcal{X}$-valued tree $\mathbf{x}$ of depth $d$ is shattered by a function class $\mathcal{F} \subseteq \{\pm 1\}^{\mathcal{X}}$ if for all $\epsilon \in \{\pm 1\}^d$, there exists $f \in \mathcal{F}$ such that $f(\mathbf{x}_t(\epsilon)) = \epsilon_t$ for all $t \in [d]$. The Littlestone dimension* $\mathrm{Ldim}(\mathcal{F}, \mathcal{X})$ *is the largest $d$ such that $\mathcal{F}$ shatters an $\mathcal{X}$-valued tree of depth $d$.*

**Definition 4.** *An $\mathcal{X}$-valued tree $\mathbf{x}$ of depth $d$ is $\alpha$-shattered by a function class $\mathcal{F} \subseteq \mathbb{R}^{\mathcal{X}}$, if there exists an $\mathbb{R}$-valued tree $\mathbf{s}$ of depth $d$ such that*

$$\forall \epsilon \in \{\pm 1\}^d, \ \exists f \in \mathcal{F} \ \text{ s.t. } \forall t \in [d], \ \epsilon_t(f(\mathbf{x}_t(\epsilon)) - \mathbf{s}_t(\epsilon)) \geq \alpha/2$$

*The tree $\mathbf{s}$ is called the witness to shattering. The fat-shattering dimension* $\mathrm{fat}_\alpha(\mathcal{F}, \mathcal{X})$ *at scale $\alpha$ is the largest $d$ such that $\mathcal{F}$ $\alpha$-shatters an $\mathcal{X}$-valued tree of depth $d$.*

With these definitions it is easy to see that $\mathrm{fat}_\alpha(\mathcal{F}, \mathcal{X}) = \mathrm{Ldim}(\mathcal{F}, \mathcal{X})$ for a binary-valued function class $\mathcal{F} \subseteq \{0, 1\}^{\mathcal{X}}$ for any $0 < \alpha \leq 1$. When $\mathcal{X}$ and/or $\mathcal{F}$ is understood from the context, we will simply write $\mathrm{fat}_\alpha$ or $\mathrm{fat}_\alpha(\mathcal{F})$ instead of $\mathrm{fat}_\alpha(\mathcal{F}, \mathcal{X})$.

Let us mention that if trees $\mathbf{x}$ are defined by constant mappings $\mathbf{x}_t(\epsilon) = x_t$, the combinatorial parameters coincide with the Vapnik-Chervonenkis dimension and with the scale-sensitive dimension $P_\gamma$. Therefore, the notions we are studying are a strict "temporal" generalizations of the VC theory.

As in statistical learning theory, the combinatorial parameters are only useful if they can be shown to capture that aspect of $\mathcal{F}$ which is important for learnability. In particular, a "size" of a function class is known to be related to complexity of learning from i.i.d. data., and the classical way to measure "size" is through a cover or a packing set. We propose the following definitions for online learning.

**Definition 5.** *A set $V$ of $\mathbb{R}$-valued trees of depth $T$ is an $\alpha$-cover (with respect to $\ell_p$-norm) of $\mathcal{F} \subseteq \mathbb{R}^{\mathcal{X}}$ on a tree $\mathbf{x}$ of depth $T$ if*

$$\forall f \in \mathcal{F}, \ \forall \epsilon \in \{\pm 1\}^T \ \exists \mathbf{v} \in V \ \text{s.t.} \quad \frac{1}{T} \sum_{t=1}^T |\mathbf{v}_t(\epsilon) - f(\mathbf{x}_t(\epsilon))|^p \leq \alpha^p$$

*The covering number* $\mathcal{N}_p(\alpha, \mathcal{F}, \mathbf{x})$ *of a function class $\mathcal{F}$ on a given tree $\mathbf{x}$ is the size of the smallest cover. Further define* $\mathcal{N}_p(\alpha, \mathcal{F}, T) = \sup_{\mathbf{x}} \mathcal{N}_p(\alpha, \mathcal{F}, \mathbf{x})$, *the maximal $\ell_p$ covering number of $\mathcal{F}$ over depth $T$ trees.*

In particular, a set $V$ of $\mathbb{R}$-valued trees of depth $T$ is a *0-cover* of $\mathcal{F} \subseteq \mathbb{R}^{\mathcal{X}}$ on a tree $\mathbf{x}$ of depth $T$ if for all $f \in \mathcal{F}$ and $\epsilon \in \{\pm 1\}^T$, there exists $\mathbf{v} \in V$ s.t. $\mathbf{v}_t(\epsilon) = f(\mathbf{x}_t(\epsilon))$. We denote by $\mathcal{N}(0, \mathcal{F}, \mathbf{x})$ the size of a smallest 0-cover on $\mathbf{x}$ and $\mathcal{N}(0, \mathcal{F}, T) = \sup_{\mathbf{x}} \mathcal{N}(0, \mathcal{F}, \mathbf{x})$. The 0-cover should not be mistaken for the size $|\{f(\mathbf{x}) : f \in \mathcal{F}\}|$ of the projection of $\mathcal{F}$ onto the tree $\mathbf{x}$, and the same care should be taken when dealing with $\alpha$-covers.

We would like to comment that while in the i.i.d. setting there is a notion of packing number that upper and lower bounds covering number, in the sequential counterpart such an analog fails.

## 5.1 A Combinatorial Upper Bound

We now relate the combinatorial parameters introduced in the previous section to the size of a cover. In the binary case ($k = 1$ below), a reader might notice a similarity of Theorem 3 to the classical results due to Sauer [24], Shelah [26] (also, Perles and Shelah), and Vapnik and Chervonenkis [32]. There are several approaches to proving what is often called the Sauer-Shelah lemma. We opt for the inductive-style proof (e.g. Alon and Spencer [4]). Dealing with trees, however, requires more work than in the VC case.

**Theorem 3.** *Let $\mathcal{F} \subseteq \{0, \ldots, k\}^{\mathcal{X}}$ be a class of functions with $\mathrm{fat}_1(\mathcal{F}) = d_1$, $\mathrm{fat}_2(\mathcal{F}) = d_2$. Then*

$$\mathcal{N}_\infty(1/2, \mathcal{F}, T) \leq \sum_{i=0}^{d_2} \binom{T}{i} k^i \leq (ekT)^{d_2}, \qquad \mathcal{N}(0, \mathcal{F}, T) \leq \sum_{i=0}^{d_1} \binom{T}{i} k^i \leq (ekT)^{d_1}.$$

Of particular interest is the case $k = 1$, when $\text{fat}_1(\mathcal{F}) = \text{Ldim}(\mathcal{F})$. Armed with Theorem 3, we can reduce the problem of bounding the size of a cover at an $\alpha$ scale by a discretization trick. For the classical case of a cover based on a set points, the discretization idea appears in [3, 22]. We now show that the covering numbers are bounded in terms of the fat-shattering dimension.

**Corollary 4.** *Suppose $\mathcal{F}$ is a class of $[-1, 1]$-valued functions on $\mathcal{X}$. Then for any $\alpha > 0$, any $T > 0$, and any $\mathcal{X}$-valued tree $\mathbf{x}$ of depth $T$,*

$$\mathcal{N}_1(\alpha, \mathcal{F}, \mathbf{x}) \leq \mathcal{N}_2(\alpha, \mathcal{F}, \mathbf{x}) \leq \mathcal{N}_\infty(\alpha, \mathcal{F}, \mathbf{x}) \leq (2eT/\alpha)^{\text{fat}_\alpha(\mathcal{F})}$$

When bounding deviations of means from expectations uniformly over the function class, the usual approach proceeds by a symmetrization argument [12] followed by passing to a cover of the function class and a union bound (e.g. [21]). Alternatively, a more refined *chaining* analysis integrates over covering at different scales (e.g. [30]). By following the same path, we are able to prove a number of similar results for our setting. Next, we present a bound similar to Massart's finite class lemma [20, Lemma 5.2]. This result will be used when integrating over different scales for the cover.

## 5.2 Finite Class Lemma and the Chaining Method

**Lemma 5.** *For any finite set $V$ of $\mathbb{R}$-valued trees of depth $T$ we have that*

$$\mathbb{E}_\epsilon \left[ \max_{\mathbf{v} \in V} \sum_{t=1}^T \epsilon_t \mathbf{v}_t(\epsilon) \right] \leq \sqrt{2 \log(|V|) \max_{\mathbf{v} \in V} \max_{\epsilon \in \{\pm 1\}^T} \sum_{t=1}^T \mathbf{v}_t(\epsilon)^2}$$

A simple consequence of the above lemma is that if $\mathcal{F} \subseteq [0, 1]^{\mathcal{X}}$ is a finite class, then for any given tree $\mathbf{x}$ we obtain a $\sqrt{2T \log(|\mathcal{F}|)}$ upper bound. If $f \in \mathcal{F}$ is associated with an "expert" (see [9]), this result combined with Theorem 2 yields a bound given by the expert's algorithm. In Section 8 we discuss this case in more detail. However, as we show next, Lemma 5 goes well beyond just finite classes and can be used to get an analog of Dudley entropy bound [10] for the online setting through a chaining argument.

**Definition 6.** The *Integrated complexity* of a function class $\mathcal{F} \subseteq [-1, 1]^{\mathcal{X}}$ is defined as

$$\mathfrak{D}_T(\mathcal{F}) = \inf_\alpha \left\{ 4T\alpha + 12 \int_\alpha^1 \sqrt{T \, \log \, \mathcal{N}_2(\delta, \mathcal{F}, T)} \, d\delta \right\}.$$

The basic idea in the proof of the following theorem is the same as in statistical learning: $\mathfrak{R}_T(\mathcal{F})$ is bounded by controlling the complexity along the chain of coverings. The argument for trees, though, is more involved than the classical case.

**Theorem 6.** *For any function class $\mathcal{F} \subseteq [-1, 1]^{\mathcal{X}}, \quad \mathfrak{R}_T(\mathcal{F}) \leq \mathfrak{D}_T(\mathcal{F})$*

# 6 Supervised Learning

In this section we study the supervised learning problem where player picks a function $f_t \in \mathbb{R}^{\mathcal{X}}$ at any time $t$ and the adversary provides input target pair $(x_t, y_t)$ and the player suffers loss $|f_t(x_t) - y_t|$. Note that if $\mathcal{F} \subseteq \{\pm 1\}^{\mathcal{X}}$ and each $y_t \in \{\pm 1\}$ then the problem boils down to binary classification problem. As we are interested in *prediction*, we allow $f_t$ to be outside of $\mathcal{F}$. Though we use the absolute loss in this section, it is easy to see that all the results hold (with modified rates) for any loss $\ell(f(x), y)$ which is such that for all $f$, $x$ and $y$, $\phi(\ell(\hat{y}, y)) \leq |\hat{y} - y| \leq \Phi(\ell(\hat{y}, y))$ where $\Phi$ and $\phi$ are monotonically increasing functions. For instance the squared loss is a classic example.

To formally define the value of the online supervised learning game, fix a set of labels $\mathcal{Y} \subseteq [-1, 1]$. Given $\mathcal{F}$, define the associated loss class, $\mathcal{F}_S = \{(x, y) \mapsto |f(x) - y| : f \in \mathcal{F}\}$. Now, the supervised game is obtained using the pair $(\mathcal{F}_S, \mathcal{X} \times \mathcal{Y})$ and we accordingly define $\mathcal{V}_T^S(\mathcal{F}) = \mathcal{V}_T(\mathcal{F}_S, \mathcal{X} \times \mathcal{Y})$. Binary classification is, of course, a special case when $\mathcal{Y} = \{\pm 1\}$ and $\mathcal{F} \subseteq \{\pm 1\}^{\mathcal{X}}$. In that case, we simply use $\mathcal{V}_T^{\text{Binary}}$ for $\mathcal{V}_T^S$.

**Proposition 7.** *For the supervised learning game played with a function class $\mathcal{F} \subseteq [-1,1]^{\mathcal{X}}$, for any $T \geq 1$*

$$\frac{1}{4\sqrt{2}} \sup_\alpha \left\{ \alpha \sqrt{T \min\{\mathrm{fat}_\alpha, T\}} \right\} \leq \frac{1}{2} \mathcal{V}_T^S(\mathcal{F})$$

$$\leq \Re_T(\mathcal{F}) \leq \mathfrak{D}_T(\mathcal{F}) \leq \inf_\alpha \left\{ 4T\alpha + 12\sqrt{T} \int_\alpha^1 \sqrt{\mathrm{fat}_\beta \log\left(\frac{2eT}{\beta}\right)} \, d\beta \right\} \tag{3}$$

**Theorem 8.** *For any function class $\mathcal{F} \subseteq [-1,1]^{\mathcal{X}}$, $\mathcal{F}$ is online learnable in the supervised setting if and only if $\mathrm{fat}_\alpha(\mathcal{F})$ is finite for any $\alpha > 0$. Moreover, if the function class is online learnable, then the value of the supervised game $\mathcal{V}_T^S(\mathcal{F})$, the Sequential Rademacher complexity $\Re(\mathcal{F})$, and the Integrated complexity $\mathfrak{D}(\mathcal{F})$ are within a multiplicative factor of $\mathcal{O}(\log^{3/2} T)$ of each other.*

**Corollary 9.** *For the binary classification game played with function class $\mathcal{F}$ we have that*

$$K_1 \sqrt{T \min\{\mathrm{Ldim}(\mathcal{F}), T\}} \leq \mathcal{V}_T^{Binary}(\mathcal{F}) \leq K_2 \sqrt{T \, \mathrm{Ldim}(\mathcal{F}) \log T}$$

*for some universal constants $K_1, K_2$. This recovers the result of [7].*

We wish to point out that lower bound of Proposition 7 also holds for "improper" supervised learning algorithms, i.e. those simply output a prediction $\hat{y}_t \in \mathcal{Y}$ rather than a function $f_t \in \mathcal{F}$. Since a proper learning strategy can always be used as an improper learning strategy, we trivially have that if class is online learnable in the supervised setting then it is improperly online learnable. Because the above mentioned property of lower bound of Proposition 7, we also have the non-trivial reverse implication: if a class is improperly online learnable in the supervised setting, it is online learnable.

### 6.1 Generic Algorithm

We shall now present a generic improper learning algorithm for the supervised setting that achieves a low regret bound whenever the function class is online learnable. For any $\alpha > 0$ define an $\alpha$-discretization of the $[-1,1]$ interval as $B_\alpha = \{-1 + \alpha/2, -1 + 3\alpha/2, \ldots, -1 + (2k+1)\alpha/2, \ldots\}$ for $0 \leq k$ and $(2k+1)\alpha \leq 4$. Also for any $a \in [-1,1]$ define $\lfloor a \rfloor_\alpha = \operatorname*{argmin}_{r \in B_\alpha} |r - a|$. For a set of functions $V \subseteq \mathcal{F}$, any $r \in B_\alpha$ and $x \in \mathcal{X}$ define $V(r, x) = \{f \in V \mid f(x) \in (r - \alpha/2, r + \alpha/2]\}$.

The algorithm proceeds by generating "experts" in a way similar to [7]. Using these experts along with exponentially weighted experts algorithm we shall provide the generic algorithm for online supervised learning.

---

**Algorithm 1** Expert $(\mathcal{F}, \alpha, 1 \leq i_1 < \ldots < i_L \leq T, Y_1, \ldots, Y_L)$

---

$V_1 \leftarrow \mathcal{F}$
**for** $t = 1$ to $T$ **do**
    $R_t(x) = \{r \in B_\alpha : \mathrm{fat}_\alpha(V_t(r, x)) = \max_{r' \in B_\alpha} \mathrm{fat}_\alpha(V_t(r', x))\}$
    For each $x \in \mathcal{X}$, let $f_t'(x) = \frac{1}{|R_t(x)|} \sum_{r \in R_t(x)} r$
    **if** $t \in \{i_1, \ldots, i_L\}$ **then**
        $\forall x \in \mathcal{X}, f_t(x) = Y_j$ where $j$ is s.t. $t = i_j$.
        Play $f_t$, receive $x_t$, and update $V_{t+1} = V_t(f_t(x_t), x_t)$
    **else**
        Play $f_t = f_t'$, receive $x_t$, and set $V_{t+1} = V_t$
    **end if**
**end for**

---

For each $L \leq \mathrm{fat}_\alpha(\mathcal{F})$ and every possible choice of $1 \leq i_1 < \ldots < i_L \leq T$ and $Y_1, \ldots, Y_L \in B_\alpha$ we generate an expert. Denote this set of experts as $E_T$. Each expert outputs a function $f_t \in \mathcal{F}$ at every round $T$. Hence each expert $e \in E_T$ can be seen as a sequence $(e_1, \ldots, e_T)$ of mappings $e_t : \mathcal{X}^{t-1} \mapsto \mathcal{F}$. The number of unique experts is $|E_T| = \sum_{L=0}^{\mathrm{fat}_\alpha} \binom{T}{L} (|B_\alpha| - 1)^L \leq \left(\frac{2T}{\alpha}\right)^{\mathrm{fat}_\alpha}$ Using an argument similar to [7], for any $f \in \mathcal{F}$ there exists $e \in E_T$ such that for any $t \in [T]$, $|f(x_t) - e(x_{1:t-1})(x_t)| \leq \alpha$ .

**Theorem 10.** *For any $\alpha > 0$ if we run the exponentially weighted experts algorithm with the set $E_T$ of experts then the expected regret of the algorithm is bounded as*

$$\mathbb{E}\left[ \sum_{t=1}^T f_t(x_t) - \inf_{f \in \mathcal{F}} \sum_{t=1}^T f(x_t) \right] \leq \alpha T + \sqrt{T \mathrm{fat}_\alpha \log\left(\frac{2T}{\alpha}\right)}$$

*Further if $\mathcal{F}$ be bounded by $1$ then by running an additional experts algorithm over the experts for discretizations over $\alpha$, we can provide regret guarantee of*

$$\mathbb{E}\left[\sum_{t=1}^{T} f_t(x_t) - \inf_{f \in \mathcal{F}} \sum_{t=1}^{T} f(x_t)\right] \leq \inf_{\alpha} \left\{ \alpha T + \sqrt{T \mathrm{fat}_\alpha \log\left(\frac{2T}{\alpha}\right)} + \sqrt{T}\left(3 + 2 \log\log\left(\frac{1}{\alpha}\right)\right)\right\}$$

# 7 Structural Results

Being able to bound complexity of a function class by a complexity of a simpler class is of great utility for proving bounds. In statistical learning theory, such structural results are obtained through properties of Rademacher averages [21, 6]. In particular, the contraction inequality due to Ledoux and Talagrand, allows one to pass from a composition of a Lipschitz function with a class to the function class itself. This wonderful property permits easy convergence proofs for a vast array of problems. We show that the notion of Sequential Rademacher complexity also enjoys many of the same properties. In Section 8, the effectiveness of the results is illustrated on a number of examples. First, we prove the contraction inequality.

**Lemma 11.** *Fix a class $\mathcal{F} \subseteq \mathbb{R}^{\mathcal{Z}}$ and a function $\phi : \mathbb{R} \times \mathcal{Z} \mapsto \mathbb{R}$. Assume, for all $z \in \mathcal{Z}$, $\phi(\cdot, z)$ is a $L$-Lipschitz function. Then $\mathfrak{R}(\phi(\mathcal{F})) \leq L \cdot \mathfrak{R}(\mathcal{F})$ where $\phi(\mathcal{F}) = \{z \mapsto \phi(f(z), z) : f \in \mathcal{F}\}$.*

The next lemma bounds the Sequential Rademacher complexity for the product of classes.

**Lemma 12.** *Let $\mathcal{F} = \mathcal{F}_1 \times \ldots \times \mathcal{F}_k$ where each $\mathcal{F}_j \subset \mathbb{R}^{\mathcal{X}}$. Also let $\phi : \mathbb{R}^k \mapsto \mathbb{R}$ be $L$-Lipschitz w.r.t. $\|\cdot\|_\infty$ norm. Then we have that $\mathfrak{R}(\phi \circ \mathcal{F}) \leq L \mathcal{O}\left(\log^{3/2}(T)\right) \sum_{j=1}^{k} \mathfrak{R}(\mathcal{F}_j)$.*

**Corollary 13.** *For a fixed binary function $b : \{\pm 1\}^k \mapsto \{\pm 1\}$ and classes $\mathcal{F}_1, \ldots, \mathcal{F}_k$ of $\{\pm 1\}$-valued functions, $\mathfrak{R}(g(\mathcal{F}_1, \ldots, \mathcal{F}_k)) \leq \mathcal{O}\left(\log^{3/2}(T)\right) \sum_{j=1}^{k} \mathfrak{R}(\mathcal{F}_j)$.*

In the next proposition, we summarize some basic properties of Sequential Rademacher complexity (see [21, 6] for the results in the i.i.d. setting):

**Proposition 14.** *Sequential Rademacher complexity satisfies the following properties: $(i)$ if $\mathcal{F} \subset \mathcal{G}$, then $\mathfrak{R}(\mathcal{F}) \leq \mathfrak{R}(\mathcal{G})$; $(ii)$ $\mathfrak{R}(\mathcal{F}) = \mathfrak{R}(\mathrm{conv}(\mathcal{F}))$; $(iii)$ $\mathfrak{R}(c\mathcal{F}) = |c|\mathfrak{R}(\mathcal{F})$ for all $c \in \mathbb{R}$; $(iv)$ If $\phi : \mathbb{R} \mapsto \mathbb{R}$ is $L$-Lipschitz, then $\mathfrak{R}(\phi(\mathcal{F})) \leq L\mathfrak{R}(\mathcal{F})$; $(v)$ For any $h$, $\mathfrak{R}(\mathcal{F} + h) = \mathfrak{R}(\mathcal{F})$ where $\mathcal{F} + h = \{f + h : f \in \mathcal{F}\}$.*

# 8 Examples and Applications

**Example: Linear Function Classes**  Suppose $\mathcal{F}_{\mathcal{W}}$ is a class consisting of linear functions $x \mapsto \langle w, x \rangle$ where the weight vector $w$ comes from some set $\mathcal{W}$, $\mathcal{F}_{\mathcal{W}} = \{x \mapsto \langle w, x \rangle : w \in \mathcal{W}\}$. Often, it is possible to find a strongly convex function $\Psi(w) \geq 0$ such that $\Psi(w) \leq \Psi_{\max} < \infty$ for all $\mathbf{w} \in \mathcal{W}$ (for example the function $\|w\|_2^2$ on any bounded subset of $\mathbb{R}^d$).

**Theorem 15.** *Let $\mathcal{W}$ be a class of weight vectors such that $0 \leq \Psi(w) \leq \Psi_{max}$ for all $w \in \mathcal{W}$. Suppose that $\Psi$ is $\sigma$-strongly convex w.r.t. a given norm $\|\cdot\|$. Then, we have, $\mathfrak{R}_T(\mathcal{F}_{\mathcal{W}}) \leq \|\mathcal{X}\|_\star \sqrt{2\Psi_{max} T/\sigma}$, where $\|\mathcal{X}\|_\star = \sup_{x \in \mathcal{X}} \|x\|_\star$, the maximum dual norm of any vector in the input space.*

The above result actually allows us to recover the $O(\sqrt{T})$ regret bounds of online mirror descent (including Zinkevich's online gradient descent) obtained in the online convex optimization literature. There, the set $\mathcal{X}$ is set of convex Lipschitz functions on a convex set $\mathcal{F}$. We interpret $f(x)$ as $x(f)$. It is easy to bound the value of the convex game by that of the linear game [2], i.e. one in which $\mathcal{X}$ is the set of linear functions. Then we directly appeal to the above theorem to bound the value of the linear game. The online convex optimization setting includes supervised learning using convex losses and linear predictors and so our theorem also proves existence of $O(\sqrt{T})$ regret algorithms in that setting.

**Example: Margin Based Regret**  We prove a general margin based mistake bound for binary classification. This shows the generality of our framework since we do not require assumptions

like convexity to bound the minimax regret. The proof of the following result uses a non-convex Lipschitz "ramp" function along with Lemma 11. As far as we know, this is the first general margin based mistake bound in the online setting for a general function class.

**Theorem 16.** *For any function class $\mathcal{F} \subset \mathbb{R}^{\mathcal{X}}$ bounded by $B$, there exists a randomized player strategy $\pi$ such that for any sequence $(x_1, y_1), \ldots, (x_T, y_T) \in (\mathcal{X} \times \{\pm 1\})^T$,*

$$\sum_{t=1}^{T} \mathbb{E}_{f_t \sim \pi_t(x_{1:t-1})} \left[ \mathbf{1} \left\{ f_t(x_t) y_t < 0 \right\} \right] \leq \inf_{\gamma > 0} \left\{ \inf_{f \in \mathcal{F}} \sum_{t=1}^{T} \mathbf{1} \left\{ f(x_t) y_t < \gamma \right\} + \frac{4}{\gamma} \mathfrak{R}_T(\mathcal{F}) + \sqrt{T} \log \log \left( \frac{B}{\gamma} \right) \right\}$$

**Example : Neural Networks and Decision Trees**   We now consider a $k$-layer 1-norm neural network. To this end let function class $\mathcal{F}_1$ be given by

$$\mathcal{F}_1 = \left\{ x \mapsto \sum_j w_j^1 x_j \ \Big| \ \|w\|_1 \leq B_1 \right\}, \text{ and } \mathcal{F}_i = \left\{ x \mapsto \sum_j w_j^i \sigma \left( f_j(x) \right) \ \Big| \ \forall j \ f_j \in \mathcal{F}_{i-1}, \|w^i\|_1 \leq B_i \right\}$$

for $2 \leq i \leq k$. The theory we have developed provides us with enough tools to control the sequential Rademacher complexity of classes like the above that are built using simpler components. The following result shows that neural networks can be learned online. A similar result, but for statistical learning, appeared in [6]. Let $\mathcal{X} \subset \mathbb{R}^d$, and $X_\infty$ be such that $\forall x \in \mathcal{X}, \|x\|_\infty \leq X_\infty$.

**Theorem 17.** *Let $\sigma : \mathbb{R} \mapsto [-1, 1]$ be $L$-Lipschitz. Then*

$$\mathfrak{R}_T(\mathcal{F}_k) \leq \left( \prod_{i=1}^{k} B_i \right) L^{k-1} X_\infty \sqrt{2T \log d}.$$

We can also prove online learnability of decision trees under appropriate restrictions on their depth and number of leaves. We skip the formal statement in the interest of space but the proof proceeds in a fashion similar to the decision tree result in [6]. The structural results enjoyed by the sequential Rademacher complexity (esp. Corollary 13) are key to making the proof work.

**Example: Transductive Learning and Prediction of Individual Sequences**   Let $\mathcal{F} \subset \mathbb{R}^{\mathcal{X}}$ and let $\widehat{\mathcal{N}}_\infty(\alpha, \mathcal{F})$ be the classical pointwise (over $\mathcal{X}$) covering number at scale $\alpha$. It is easy to verify that $N_\infty(\alpha, \mathcal{F}, T) \leq \widehat{\mathcal{N}}_\infty(\alpha, \mathcal{F})$ for all $T$. This simple observation can be applied in several situations. First, consider *transductive learning*, where the set $\mathcal{X} = \{z_1\}_{i=1}^n$ is a finite set. To ensure online learnability, it is sufficient to consider an assumption on the dependence of $\widehat{\mathcal{N}}_\infty(\alpha, \mathcal{F})$ on $\alpha$. An obvious example of such a class is a VC-type class with $\widehat{\mathcal{N}}_\infty(\alpha, \mathcal{F}) \leq (c/\alpha)^d$ for some $c$ which can depend on $n$. Assuming that $\mathcal{F} \subset [0, 1]^{\mathcal{X}}$, the value of the game is upper bounded by $2\mathfrak{D}_T(\mathcal{F}) \leq 4\sqrt{dT \log c}$. In particular, for binary prediction, using the Sauer-Shelah lemma ensures that the value of the game is at most $4\sqrt{dT \log(eT)}$, matching the result of [15] up to a constant 2.

In the context of prediction of individual sequences, Cesa-Bianchi and Lugosi [8] proved upper bounds in terms of the (classical) Rademacher complexity and the (classical) Dudley integral. The particular assumption made in [8] is that experts are *static*. Formally, we define static experts as mappings $f : \{1, \ldots, T\} \mapsto [0, 1]$, and let $\mathcal{F}$ denote a class of such experts. Defining $\mathcal{X} = \{1, \ldots, T\}$ puts us in the setting considered earlier with $n = T$. We immediately obtain $4\sqrt{dT \log(eT)}$, matching the results on [8, p. 1873]. For the case of a finite number of experts, clearly $\widehat{\mathcal{N}}_\infty \leq N$ which gives the classical $O(\sqrt{T \log N})$ bound [9].

**Example: Isotron**   Recently, Kalai and Sastry [16] introduced a method called *Isotron* for learning Single Index Models (SIM), which generalize linear and logistic regression, generalized linear models, and classification by linear threshold functions. A natural open question posed by the authors is whether there is an online variant of Isotron. Before even attempting a quest for such an algorithm, we can ask a more basic question: is the (Idealized) SIM problem even learnable in the online framework? We answer the question in positive with the tools we have developed by proving that the following class (with $\mathcal{X}$ a Euclidean ball in $\mathbb{R}^d$ and $\mathcal{Y} = [-1, 1]$) is learnable:

$$\mathcal{H} = \{f(x, y) = (y - u(\langle w, x \rangle))^2 \mid u : [-1, 1] \mapsto [-1, 1] \text{ is non-decreasing 1-Lipschitz }, \|w\|_2 \leq 1\} \quad (4)$$

where $u$ and $w$ range over the possibilities. Using the machinery we developed, it is not hard to show that the class $\mathcal{H}$ is online learnable in the supervised setting. Moreover, $\mathcal{V}_T(\mathcal{H}, \mathcal{X} \times \mathcal{Y}) = O(\sqrt{T} \log^{3/2} T)$.

# References

[1] J. Abernethy, A. Agarwal, P. Bartlett, and A. Rakhlin. A stochastic view of optimal regret through minimax duality. In *Proceedings of the 22nd Annual Conference on Learning Theory*, 2009.

[2] J. Abernethy, P. L. Bartlett, A. Rakhlin, and A. Tewari. Optimal strategies and minimax lower bounds for online convex games. In *COLT*, pages 414–424, 2008.

[3] N. Alon, S. Ben-David, N. Cesa-Bianchi, and D. Haussler. Scale-sensitive dimensions, uniform convergence, and learnability. *Journal of the ACM*, 44:615–631, 1997.

[4] N. Alon and J. Spencer. *The Probabilistic Method*. John Wiley & Sons, 2nd edition, 2000.

[5] P. L. Bartlett, P. M. Long, and R. C. Williamson. Fat-shattering and the learnability of real-valued functions. *Journal of Computer and System Sciences*, 52(3):434–452, 1996. (special issue on COLT'94).

[6] P. L. Bartlett and S. Mendelson. Rademacher and gaussian complexities: risk bounds and structural results. *J. Mach. Learn. Res.*, 3:463–482, 2003.

[7] S. Ben-David, D. Pal, and S. Shalev-Shwartz. Agnostic online learning. In *COLT*, 2009.

[8] N. Cesa-Bianchi and G. Lugosi. On prediction of individual sequences. *A. of S.*, pages 1865–1895, 1999.

[9] N. Cesa-Bianchi and G. Lugosi. *Prediction, Learning, and Games*. Cambridge University Press, 2006.

[10] R. M. Dudley. The sizes of compact subsets of Hilbert space and continuity of Gaussian processes. *Journal of Functional Analysis*, 1(3):290–330, 1967.

[11] R. M. Dudley. *Uniform Central Limit Theorems*. Cambridge University Press, 1999.

[12] E. Giné and J. Zinn. Some limit theorems for empirical processes. *Ann. of Prob.*, 12(4):929–989, 1984.

[13] J. Hannan. Approximation to Bayes risk in repeated play. *Contr. to Theo. of Games*, 3:97–139, 1957.

[14] D. Haussler. Decision theoretic generalizations of the PAC model for neural net and other learning applications. *Information and Computation*, 100(1):78–150, 1992.

[15] S. M. Kakade and A. Kalai. From batch to transductive online learning. In *NIPS*, 2005.

[16] A. Tauman Kalai and R. Sastry. The isotron algorithm: High-dimensional isotonic regression. In *Proceedings of the 22th Annual Conference on Learning Theory*, 2009.

[17] M. J. Kearns and R. E. Schapire. Efficient distribution-free learning of probabilistic concepts. *Journal of Computer and System Sciences*, 48(3):464–497, 1994.

[18] V. Koltchinskii and D. Panchenko. Rademacher processes and bounding the risk of function learning. *High Dimensional Probability II*, 47:443–459, 2000.

[19] N. Littlestone. Learning quickly when irrelevant attributes abound: A new linear-threshold algorithm. *Machine Learning*, 2(4):285–318, 04 1988.

[20] P. Massart. Some applications of concentration inequalities to statistics. *Annales de la Faculté des Sciences de Toulouse*, IX(2):245–303, 2000.

[21] S. Mendelson. A few notes on statistical learning theory. In *MLSS 2002*, pages 1–40. 2003.

[22] S. Mendelson and R. Vershynin. Entropy and the combinatorial dimension. *Inventiones mathematicae*, 152:37–55, 2003.

[23] D. Pollard. *Empirical Processes: Theory and Applications*, volume 2. Hayward, CA, 1990.

[24] N. Sauer. On the density of families of sets. *J. Combinatorial Theory*, 13:145–147, 1972.

[25] S. Shalev-Shwartz and Y. Singer. Convex repeated games and fenchel duality. In *NIPS*, pages 1265–1272. MIT Press, Cambridge, MA, 2007.

[26] S. Shelah. A combinatorial problem: Stability and order for models and theories in infinitary languages. *Pac. J. Math*, 4:247–261, 1972.

[27] K. Sridharan and A. Tewari. Convex games in banach spaces. In *COLT*, 2010.

[28] A. W. Van Der Vaart and J. A. Wellner. *Weak Convergence and Empirical Processes : With Applications to Statistics*. Springer Series, March 1996.

[29] L. Valiant. A theory of the learnable. *Communications of the ACM*, 27(11):1134–1142, 1984.

[30] S.A. van de Geer. *Empirical Processes in M-Estimation*. Cambridge University Press, 2000.

[31] V. N. Vapnik. *Estimation of Dependences Based on Empirical Data (Springer Series in Statistics)*. Springer-Verlag New York, Inc., Secaucus, NJ, USA, 1982.

[32] V. N. Vapnik and A. Ya. Chervonenkis. On the uniform convergence of relative frequencies of events to their probabilities. *Theory of Probability and its Applications*, 16(2):264–280, 1971.

[33] M. Zinkevich. Online convex programming and generalized infinitesimal gradient ascent. In *ICML*, pages 928–936, 2003.

